# Mixtures of Gaussian Processes

**Volker Tresp**

Siemens AG, Corporate Technology, Department of Neural Computation
Otto-Hahn-Ring 6, 81730 München, Germany
*Volker.Tresp@mchp.siemens.de*

## Abstract

We introduce the mixture of Gaussian processes (MGP) model which is useful for applications in which the optimal bandwidth of a map is input dependent. The MGP is derived from the mixture of experts model and can also be used for modeling general conditional probability densities. We discuss how Gaussian processes —in particular in form of Gaussian process classification, the support vector machine and the MGP model— can be used for quantifying the dependencies in graphical models.

## 1   Introduction

Gaussian processes are typically used for regression where it is assumed that the underlying function is generated by one infinite-dimensional Gaussian distribution (i.e. we assume a Gaussian prior distribution). In Gaussian process regression (GPR) we further assume that output data are generated by additive Gaussian noise, i.e. we assume a Gaussian likelihood model. GPR can be generalized by using likelihood models from the exponential family of distributions which is useful for classification and the prediction of lifetimes or counts. The support vector machine (SVM) is a variant in which the likelihood model is not derived from the exponential family of distributions but rather uses functions with a discontinuous first derivative. In this paper we introduce another generalization of GPR in form of the mixture of Gaussian processes (MGP) model which is a variant of the well known mixture of experts (ME) model of Jacobs *et al.* (1991). The MGP model allows Gaussian processes to model general conditional probability densities. An advantage of the MGP model is that it is fast to train, if compared to the neural network ME model. Even more interesting, the MGP model is one possible approach of addressing the problem of input-dependent bandwidth requirements in GPR. Input-dependent bandwidth is useful if either the complexity of the map is input dependent —requiring a higher bandwidth in regions of high complexity— or if the input data distribution is input dependent. In the latter case, one would prefer Gaussian processes with a higher bandwidth in regions with many data points and a lower bandwidth in regions with lower data density. If GPR models with different bandwidths are used, the MGP approach allows the system to self-organize by locally selecting the GPR model with the appropriate optimal bandwidth.

Gaussian process classifiers, the support vector machine and the MGP can be used to model the local dependencies in graphical models. Here, we are mostly interested in the case that the dependencies of a set of variables $y$ is modified via Gaussian processes by a set of exogenous variables $x$. As an example consider a medical domain in which a Bayesian network of discrete variables $y$ models the dependencies between diseases and symptoms and

where these dependencies are modified by exogenous (often continuous) variables $x$ representing quantities such as the patient's age, weight or blood pressure. Another example would be collaborative filtering where $y$ might represent a set of goods and the correlation between customer preferences is modeled by a dependency network (another example of a graphical model). Here, exogenous variables such as income, gender and social status might be useful quantities to modify those dependencies.

The paper is organized as follows. In the next section we briefly review Gaussian processes and their application to regression. In Section 3 we discuss generalizations of the simple GPR model. In Section 4 we introduce the MGP model and present experimental results. In Section 5 we discuss Gaussian processes in context with graphical models. In Section 6 we present conclusions.

## 2  Gaussian Processes

In Gaussian Process Regression (GPR) one assumes that *a priori* a function $f(x)$ is generated from an infinite-dimensional Gaussian distribution with zero mean and covariance $K(x, x_k) = cov(f(x), f(x_k))$ where $K(x, x_k)$ are positive definite kernel functions. In this paper we will only use Gaussian kernel functions of the form

$$K(x, x_k) = A \exp\left(-\frac{\|x - x_k\|^2}{2s^2}\right)$$

with scale parameter $s$ and amplitude $A$. Furthermore, we assume a set of $N$ training data $D = \{(x_k, y_k)\}_{k=1}^{N}$ where targets are generated following a normal distribution with variance $\sigma^2$ such that

$$P(y|f(x)) \propto \exp\left(-\frac{1}{2\sigma^2}(f(x) - y)^2\right). \tag{1}$$

The expected value $\hat{f}(x)$ to an input $x$ given the training data is a superposition of the kernel functions of the form

$$\hat{f}(x) = \sum_{k=1}^{N} w_k K(x, x_k). \tag{2}$$

Here, $w_k$ is the weight on the $k$-th kernel. Let $K$ be the $N \times N$ Gram matrix with $(K)_{k,j} = cov(f(x_k), f(x_j))$. Then we have the relation $f^m = Kw$ where the components of $f^m = (f(x_1), \ldots, f(x_N))'$ are the values of $f$ at the location of the training data and $w = (w_1, \ldots, w_N)'$. As a result of this relationship we can either calculate the optimal $w$ or we can calculate the optimal $f^m$ and then deduce the corresponding $w$-vector by matrix inversion. The latter approach is taken in this paper. Following the assumptions, the optimal $f^m$ minimizes the cost function

$$\frac{1}{2}(f^m)'K^{-1}f^m + \frac{1}{2\sigma^2}(f^m - y)'(f^m - y) \tag{3}$$

such that

$$\hat{f}^m = K(K + \sigma^2 I)^{-1}y.$$

Here $y = (y_1, \ldots, y_N)'$ is the vector of targets and $I$ is the $N$-dimensional unit matrix.

## 3  Generalized Gaussian Processes and the Support Vector Machine

In generalized Gaussian processes the Gaussian prior assumption is maintained but the likelihood model is now derived from the exponential family of distributions. The most

important special cases are two-class classification

$$P(y = 1|f(x)) = \frac{1}{1 + \exp(-f(x))}$$

and multiple-class classification. Here, $y$ is a discrete variable with $C$ states and

$$P(y = i|f_1(x), \ldots, f_C(x)) = \frac{\exp(f_i(x))}{\sum_{j=1}^{C} \exp(f_j(x))}. \tag{4}$$

Note, that for multiple-class classification $C$ Gaussian processes $f_1(x), \ldots, f_C(x)$ are used. Generalized Gaussian processes are discusses in Tresp (2000). The special case of classification was discussed by Williams and Barber (1998) from a Bayesian perspective. The related smoothing splines approaches are discussed in Fahrmeir and Tutz (1994). For generalized Gaussian processes, the optimization of the cost function is based on an iterative Fisher scoring procedure.

Incidentally, the support vector machine (SVM) can also be considered to be a generalized Gaussian process model with

$$P(y|f(x)) \propto \exp\left(-const(1 - yf(x))_+\right).$$

Here, $y \in \{-1, 1\}$, the operation $()_+$ sets all negative values equal to zero and *const* is a constant (Sollich (2000)).[1] The SVM cost function is particularly interesting since due to its discontinuous first derivative, many components of the optimal weight vector $w$ are zero, i.e. we obtain sparse solutions.

## 4 Mixtures of Gaussian Processes

GPR employs a global scale parameter $s$. In many applications it might be more desirable to permit an input-dependent scale parameter: the complexity of the map might be input dependent or the input data density might be nonuniform. In the latter case one might want to use a smaller scale parameter in regions with high data density. This is the main motivation for introducing another generalization of the simple GPR model, the mixture of Gaussian processes (MGP) model, which is a variant of the mixture of experts model of Jacobs *et al.* (1991). Here, a set of GPR models with different scale parameters is used and the system can autonomously decide which GPR model is appropriate for a particular region of input space. Let $F^\mu(x) = \{f_1^\mu(x), \ldots, f_M^\mu(x)\}$ denote this set of $M$ GPR models. The state of a discrete $M$-state variable $z$ determines which of the GPR models is active for a given input $x$. The state of $z$ is estimated by an $M$-class classification Gaussian process model with

$$P(z = i|F^z(x)) = \frac{\exp(f_i^z(x))}{\sum_{j=1}^{M} \exp(f_j^z(x))}$$

where $F^z(x) = \{f_1^z(x), \ldots, f_M^z(x)\}$ denotes a second set of $M$ Gaussian processes. Finally, we use a set of $M$ Gaussian processes $F^\sigma(x) = \{f_1^\sigma(x), \ldots, f_M^\sigma(x)\}$ to model the input-dependent noise variance of the GPR models. The likelihood model given the state of $z$

$$P(y|z, F^\mu(x), F^\sigma(x)) = G\left(y; f_z^\mu(x), \exp(2f_z^\sigma(x))\right)$$

is a Gaussian centered at $f_z^\mu(x)$ and with variance $(\exp(2f_z^\sigma(x)))$. The exponential is used to ensure positivity. Note that $G(a; b, c)$ is our notation for a Gaussian density with mean $b$, variance $c$, evaluated at $a$. In the remaining parts of the paper we will not denote the

dependency on the Gaussian processes explicitly, e.g we will write $P(y|z,x)$ instead of $P(y|z, F^\mu(x), F^\sigma(x))$. Since $z$ is a latent variable we obtain with

$$P(y|x) = \sum_{i=1}^{M} P(z=i|x) \, G\left(y; f_i^\mu(x), \exp(2f_i^\sigma(x))\right) \quad E(y|x) = \sum_{i=1}^{M} P(z=i|x) \, f_i^\mu(x)$$

the well known mixture of experts network of Jacobs *et al* (1991) where the $f_i^\mu(x)$ are the (Gaussian process) experts and $P(z=i|x)$ is the gating network. Figure 2 (left) illustrates the dependencies in the GPR model.

### 4.1 EM Fisher Scoring Learning Rules

Although *a priori* the functions $f$ are Gaussian distributed, this is not necessarily true –in contrast to simple GPR in Section 2– for the posterior distribution due to the nonlinear nature of the model. Therefore one is typically interested in the minimum of the negative logarithm of the posterior density

$$-\sum_{k=1}^{N} \log \sum_{i=1}^{M} P(z=i|x_k) \, G\left(y_k; f_i^\mu(x_k), \exp(2f_i^\sigma(x_k))\right)$$

$$+\frac{1}{2}\sum_{i=1}^{M}(f_i^{z,m})'(\Sigma_i^{z,m})^{-1}f_i^{z,m} + \frac{1}{2}\sum_{i=1}^{M}(f_i^{\mu,m})'(\Sigma_i^{\mu,m})^{-1}f_i^{\mu,m} + \frac{1}{2}\sum_{i=1}^{M}(f_i^{\sigma,m})'(\Sigma_i^{\sigma,m})^{-1}f_i^{\sigma,m}.$$

The superscript $m$ denotes the vectors and matrices defined at the measurement point, e.g. $f_i^{\mu,m} = (f_i^\mu(x_1), \ldots, f_i^\mu(x_N))'$. In the E-step, based on the current estimates of the Gaussian processes at the data points, the state of the latent variable is estimated as

$$\hat{P}(z=i|x_k,y_k) = \frac{\hat{P}(z=i|x_k) \, G\left(y_k; \hat{f}_i^\mu(x_k), \exp(2\hat{f}_i^\sigma(x_k))\right)}{\sum_{j=1}^{M} \hat{P}(z=j|x_k) \, G\left(y_k; \hat{f}_j^\mu(x_k), \exp(2\hat{f}_j^\sigma(x_k))\right)}.$$

In the M-step, based on the E-step, the Gaussian processes at the data points are updated. We obtain

$$\hat{f}_i^{\mu,m} = \Sigma_i^{\mu,m}\left(\Sigma_i^{\mu,m} + \Psi_i^{\mu,m}\right)^{-1} y^m$$

where $\Psi_i^{\mu,m}$ is a diagonal matrix with entries

$$(\Psi_i^{\mu,m})_{kk} = \exp(2\hat{f}_i^\sigma(x_k))/\hat{P}(z=i|x_k,y_k).$$

Note, that data with a small $\hat{P}(z=i|x_k,y_k)$ obtain a small weight. To update the other Gaussian processes iterative Fisher scoring steps have to be used as shown in the appendix.

There is a serious problem with overtraining in the MGP approach. The reason is that the GPR model with the highest bandwidth tends to obtain the highest weight in the E-step since it provides the best fit to the data. There is an easy fix for the MGP: For calculating the responses of the Gaussian processes at $x_k$ in the E-step we use all training data *except* $(x_k, y_k)$. Fortunately, this calculation is very cheap in the case of Gaussian processes since for example

$$\tilde{f}_i^\mu(x_k) = y_k - \frac{y_k - \hat{f}_i^\mu(x_k)}{1 - S_{i,kk}}$$

where $\tilde{f}_i^\mu(x_k)$ denotes the estimates at the training data point $x_k$ not using $(x_k, y_k)$. Here, $S_{i,kk}$ is the $k$-th diagonal element of $S_i = \Sigma_i^{\mu,m}\left(\Sigma_i^{\mu,m} + \Psi_i^{\mu,m}\right)^{-1}$.[2]

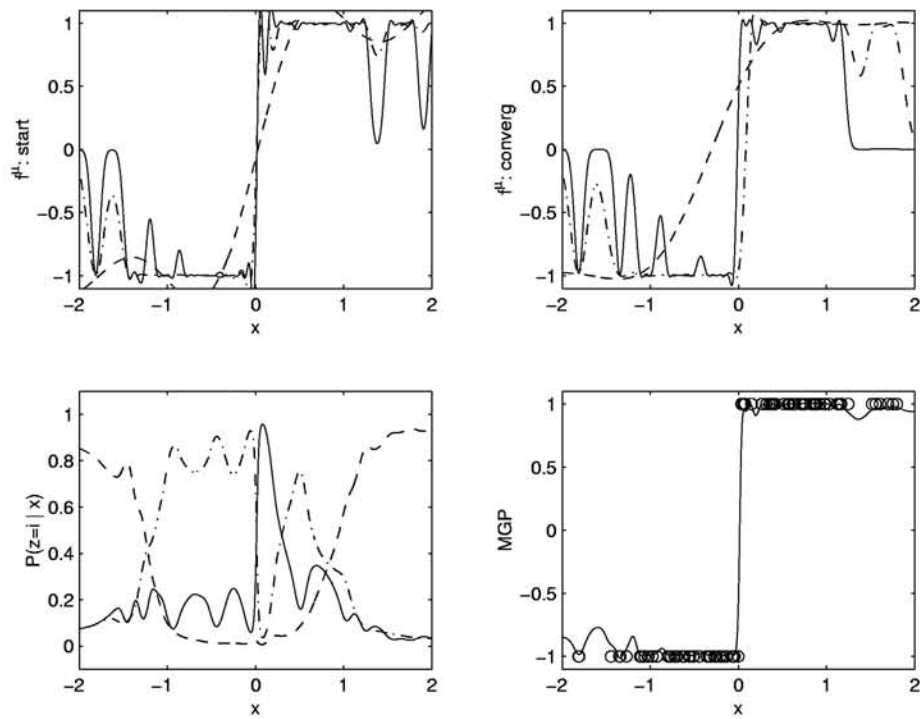

Figure 1: The input data are generated from a Gaussian distribution with unit variance and mean 0. The output data are generated from a step function ($o$, bottom right). The top left plot shows the map formed by three GPR models with different bandwidths. As can be seen no individual model achieves a good map. Then a MGP model was trained using the three GPR models. The top right plot shows the GPR models after convergence. The bottom left plot shows $P(z = i|x)$. The GPR model with the highest bandwidth models the transition at zero, the GPR model with an intermediate bandwidth models the intermediate region and the GPR model with the lowest bandwidth models the extreme regions. The bottom right plot shows the data $o$ and the fit obtained by the complete MGP model which is better than the map formed by any of the individual GPR models.

## 4.2   Experiments

Figure 1 illustrates how the MGP divides up a complex task into subtasks modeled by the individual GPR models (see caption). By dividing up the task, the MGP model can potentially achieve a performance which is better than the performance of any individual model. Table 1 shows results from artificial data sets and real world data sets. In all cases, the performance of the MGP is better than the mean performance of the GPR models and also better than the performance of the mean (obtained by averaging the predictions of all GPR models).

## 5   Gaussian Processes for Graphical Models

Gaussian processes can be useful models for quantifying the dependencies in Bayesian networks and dependency networks (the latter were introduced in Hofmann and Tresp, 1998, Heckerman *et al.*, 2000), in particular when parent variables are continuous quantities. If the child variable is discrete, Gaussian process classification or the SVM are appropriate models whereas when the child variable is continuous, the MGP model can be employed as a general conditional density estimator. Typically one would require that the continuous input variables to the Gaussian process systems $x$ are known. It might therefore be

Table 1: The table shows results using artificial and real data sets of size $N = 100$ using $M = 10$ GPR models. The data set ART is generated by adding Gaussian noise with a standard deviation of 0.2 to a map defined by 5 normalized Gaussian bumps. *numin* is the number of inputs. The bandwidth $s$ was generated randomly between 0 and max. $s$. Furthermore, *mean perf.* is the mean squared test set error of all GPR networks and *perf. of mean* is the mean squared test set error achieved by simple averaging the predictions. The last column shows the performance of the MGP.

| Data | numin | max. $s$ | mean perf. | perf. of mean | MGP |
|------|-------|----------|-----------|---------------|-----|
| ART | 1 | 1 | 0.0167 | 0.0080 | 0.0054 |
| ART | 2 | 3 | 0.0573 | 0.0345 | 0.0239 |
| ART | 5 | 6 | 0.1994 | 0.1383 | 0.0808 |
| ART | 10 | 10 | 0.1670 | 0.1135 | 0.0739 |
| ART | 20 | 20 | 0.1716 | 0.1203 | 0.0662 |
| HOUSING | 13 | 10 | 0.4677 | 0.3568 | 0.2634 |
| BUPA | 6 | 20 | 0.9654 | 0.9067 | 0.8804 |
| DIABETES | 8 | 40 | 0.8230 | 0.7660 | 0.7275 |
| WAVEFORM | 21 | 40 | 0.6295 | 0.5979 | 0.4453 |

useful to consider those as exogenous variables which modify the dependencies in a graphical model of $y$-variables as shown in Figure 2 (right). As an example consider a medical domain in which a Bayesian network of discrete variables $y$ models the dependencies between diseases and symptoms and where these dependencies are modified by exogenous (often continuous) variables $x$ representing quantities such as the patient's age, weight or blood pressure. Another example would be collaborative filtering where $y$ might represent a set of goods and the correlation between customer preferences is modeled by a dependency network as in Heckerman *et al.* (2000). Here, exogenous variables such as income, gender and social status might be useful quantities to modify those correlations. Note, that the GPR model itself can also be considered to be a graphical model with dependencies modeled as Gaussian processes (compare Figure 2).

Readers might also be interested in the related and independent paper by Friedman and Nachman (2000) in which those authors used GPR systems (not in form of the MGP) to perform structural learning in Bayesian networks of continuous variables.

# 6 Conclusions

We demonstrated that Gaussian processes can be useful building blocks for forming complex probabilistic models. In particular we introduced the MGP model and demonstrated how Gaussian processes can model the dependencies in graphical models.

# 7 Appendix

For $f^z$ and $f^\sigma$ the mode estimates are found by iterating Newton-Raphson equations $\hat{f}^{(l+1)} = \hat{f}^{(l)} - \tilde{H}^{-1}(l)J(l)$ where $J(l)$ is the Jacobian and $\tilde{H}(l)$ the Hessian matrix for which certain interactions are ignored. One obtains for ($l = 1, 2, \ldots$) the following update equations.

$$\hat{f}_i^{z,m,(l+1)} = \Sigma_i^{z,m} \left( \Sigma_i^{z,m} + \Psi_i^{z,m,(l)} \right)^{-1} \left( \Psi_i^{z,m,(l)} d_i^{z,m,(l)} + \hat{f}_i^{z,m,(l)} \right) \quad \text{where}$$

$$d_i^{z,m,(l)} = \left( \hat{P}(z = i|x_k, y_k) - \hat{P}^{(l)}(z = i|x_k) \right)_{k=1}^N,$$

$$\Psi_i^{z,m,(l)} = diag \left( [\hat{P}^{(l)}(z = i|x_k)(1 - \hat{P}^{(l)}(z = i|x_k))]^{-1} \right)_{k=1}^N.$$

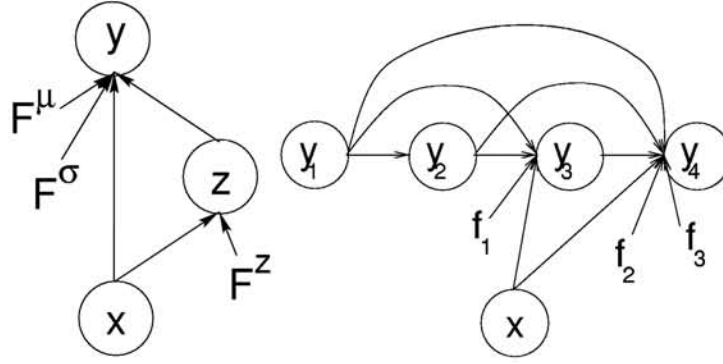

Figure 2: Left: The graphical structure of an MGP model consisting of the discrete latent variable $z$, the continuous variable $y$ and input variable $x$. The probability density of $z$ is dependent on the Gaussian processes $F^z$. The probability distribution of $y$ is dependent on the state of $z$ and of the Gaussian processes $F^\mu$, $F^\sigma$. Right: An example of a Bayesian network which contains the variables $y_1, y_2, y_3, y_4$. Some of the dependencies are modified by $x$ via Gaussian processes $f_1, f_2, f_3$.

Similarly,

$$\hat{f}_i^{\sigma,m,(l+1)} = \Sigma_i^{\sigma,m} \left( \Sigma_i^{\sigma,m} + \Delta_i^{\sigma,m,(l)} \right)^{-1} \left( \frac{1}{2}e - \psi_i^{\sigma,m,(l)} + \hat{f}_i^{\sigma,m,(l)} \right)$$

where $e$ is an $N$-dimensional vector of ones and

$$\psi_i^{\sigma,m,(l)} = \left( \frac{\exp(2\hat{f}_i^{\sigma,(l)}(x_k))}{2(\hat{f}_i^\mu(x_k) - y_k)^2} \right)_{k=1}^N$$

$$\Delta_i^{\sigma,m,(l)} = diag \left( \frac{\exp(2\hat{f}_i^{\sigma,(l)}(x_k))}{2P(z=i|x_k,y_k)(\hat{f}_i^\mu(x_k) - y_k)^2} \right)_{k=1}^N .$$

## Footnotes

[1]Properly normalizing the conditional probability density is somewhat tricky and is discussed in detail in Sollich (2000).

[2] See Hofmann (2000) for a discussion of the convergence of this type of algorithms.

## References

[1] Jacobs, R. A., Jordan, M. I., Nowlan, S. J., Hinton, J. E. (1991). Adaptive Mixtures of Local Experts, *Neural Computation, 3.*

[2] Tresp, V. (2000). The Generalized Bayesian Committee Machine. *Proceedings of the Sixth ACM SIGKDD International Conference on Knowledge Discovery and Data Mining, KDD-2000.*

[3] Williams, C. K. I., Barber. D. (1998). Bayesian Classification with Gaussian Processes, *IEEE Transactions on Pattern Analysis and Machine Intelligence,* 20(12).

[4] Fahrmeir, L., Tutz, G. (1994) *Multivariate Statistical Modeling Based on Generalized Linear Models,* Springer.

[5] Sollich, P. (2000). Probabilistic Methods for Support Vector Machines. In Solla, S. A., Leen, T. K., Müller, K.-R. (Eds.), *Advances in Neural Information Processing Systems 12,* MIT Press.

[6] Hofmann R. (2000). *Lernen der Struktur nichtlinearer Abhängingkeiten mit graphischen Modellen.* PhD Dissertation.

[7] Hofmann, R., Tresp, V. (1998). Nonlinear Markov Networks for Continuous Variables. In Jordan, M. I., Kearns, M. S., Solla, S. A., (Eds.), *Advances in Neural Information Processing Systems 10,* MIT Press.

[8] Heckerman, D., Chickering, D., Meek, C., Rounthwaite, R., Kadie C. (2000). Dependency Networks for Inference, Collaborative Filtering, and Data Visualization.. *Journal of Machine Learning Research,* 1.

[9] Friedman, N., Nachman, I. (2000). Gaussian Process Networks. In Boutilier, C., Goldszmidt, M., (Eds.), *Proc. Sixteenth Conf. on Uncertainty in Artificial Intelligence (UAI).*
